# Products of "Edge-perts"

**Peter Gehler**
Max Planck Institute for Biological Cybernetics
Spemannstraße 38, 72076 Tübingen, Germany
`pgehler@tuebingen.mpg.de`

**Max Welling**
Department of Computer Science
University of California Irvine
`welling@ics.uci.edu`

## Abstract

Images represent an important and abundant source of data. Understanding their statistical structure has important applications such as image compression and restoration. In this paper we propose a particular kind of probabilistic model, dubbed the "products of edge-perts model" to describe the structure of wavelet transformed images. We develop a practical denoising algorithm based on a single edge-pert and show state-of-the-art denoising performance on benchmark images.

## 1 Introduction

Images, when represented as a collection of pixel values, exhibit a high degree of redundancy. Wavelet transforms, which capture most of the second order dependencies, form the basis of many successful image processing applications such as image compression (e.g. JPEG2000) or image restoration (e.g. wavelet coring). However, the higher order dependencies can not be filtered out by these linear transforms. In particular, the absolute values of neighboring wavelet coefficients (but not their signs) are mutually dependent. This kind of dependency is caused by the presence of edges that induce clustering of wavelet activity. Our philosophy is that by modelling this clustering effect we can potentially improve the performance of some important image processing tasks.

Our model builds on earlier work in the image processing literature. In particular, the PoEdges models that we discuss in this paper can be viewed as generalizations of the models proposed in [1] and [2]. The state-of-art in this area is the joint model discussed in [3] based on the "Gaussian scale mixture" model (GSM). While the GSM falls in the category of directed graphical models and has a top-down structure, the PoEdges model is best classified as an (undirected) Markov random field model and follows bottom-up semantics.

The main contributions of this paper are 1) a new model to describe the higher order statistical dependencies among wavelet coefficients (section 2), 2) an efficient estimation procedure to fit the parameters of a single edge-pert model and a new technique to estimate the wavelet coefficients that participate in each such (local) model (section 3.1) and 3) a new "iterated Wiener denoising algorithm" (section 3.2). In section 4 we report on a number of experiments to compare performance of our algorithm with several methods in the literature and with the GSM-based method in particular.

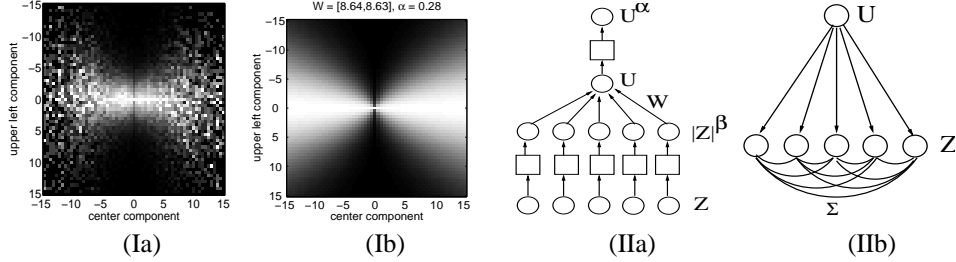

|(Ia)|(Ib)|(IIa)|(IIb)|

Figure 1: Estimated (Ia) and modelled (Ib) conditional distribution of a wavelet coefficient given its upper left neighbor. The statistics were collected from the vertical subband at the lowest level of a Haar filter wavelet decomposition of the "Lena" image. Note that the "bow-tie" dependencies are captured by the PoEdges model. (IIa) Bottom up network interpretation of "products of edge-perts" model. (IIb) Top-down generative Gaussian scale mixture model.

## 2 "Product of Edge-perts"

It has long been recognized in the image processing community that wavelet transforms form an excellent basis for representation of images. Within the class of *linear* transforms, it represents a compromise between many conflicting but desirable properties of image representation such as multi-scale and multi-orientation representation, locality both in space and frequency, and orthogonality resulting in decorrelation. A particularly suitable wavelet transform which forms the basis of the best denoising algorithms today is the over-complete steerable wavelet pyramid [4] freely downloadable from *http://www.cns.nyu.edu/~lcv/software.html*. In our experiments we have confirmed that the best results were obtained using this wavelet pyramid.

In the following we will describe a model for the statistical dependencies between wavelet coefficients. This model was inspired by recent studies of these dependencies (see e.g. [1, 5]). It also represents a generalization of the bivariate Laplacian model proposed in [2]. The probability distribution of the "product of edge-pert" model (PoEdges) over the wavelet coefficients $\mathbf{z}$ has the following form,

$$P(\mathbf{z}) = \frac{1}{Z} \exp\Big[ -\sum_i \Big(\sum_j W_{ij}|\hat{\mathbf{a}}_j^T \mathbf{z}|^{\beta_j}\Big)^{\alpha_i}\Big], \quad \beta_j > 0, \ \alpha_i \in (0,1], \ W_{ij} \geq 0$$

where the normalization constant $Z$ depends on all the parameters in the model $\{W_{ij}, \hat{\mathbf{a}}_j, \beta_j, \alpha_i\}$ and where $\hat{\mathbf{a}}$ indicates an unit-length vector.

In figure 2 we show the effect of changing some parameters for a single edge-pert model (i.e. set $i = 1$ in Eqn.1 above). The parameters $\{\beta_j\}$ control the shape of the contours: for $\beta = 2$ we have elliptical contours, for $\beta = 1$ the contours are straight lines while for $\beta < 1$ the contours curve inwards. The parameters $\{\alpha_i\}$ control the rate at which the distribution decays, i.e. the distance between iso-probability contours. The unit vectors $\{\hat{\mathbf{a}}_i\}$ determine the orientation of basis vectors. If the $\{\hat{\mathbf{a}}_i\}$ are axis-aligned (as in figure 2), the distribution is symmetric w.r.t. reflections of any subset of the $\{z_i\}$ in the origin, which implies that the wavelet coefficients are necessarily decorrelated (although higher order dependencies may still remain). Finally, the weights $\{W_{ij}\}$ model the scale (inverse variance) of the wavelet coefficients. We mention that it is possible to entertain a larger number of bases vectors than wavelet coefficients (a so-called "over-complete basis"), which seems appropriate for some of the empirical joint histograms shown in [1].

This model describes two important statistical properties which have been observed for wavelet coefficients: 1) its marginal distributions $p(z_i)$ are peaked and have heavy tails (high kurtosis) and 2) the conditional distributions $p(z_i|z_j)$ display "bow-tie" dependencies which are indicative of clustering of wavelet coefficients (neighboring wavelet coefficient

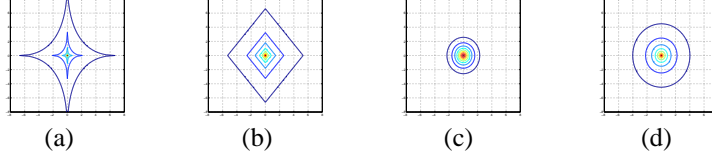

| (a) | (b) | (c) | (d) |

Figure 2: Contour plots for a single edge-pert model with (a) $\beta_{1,2} = 0.5$, $\alpha = 0.5$, (b) $\beta_{1,2} = 1$, $\alpha = 0.5$, (c) $\beta_{1,2} = 2$, $\alpha = 0.5$, (d) $\beta_{1,2} = 2$, $\alpha = 0.3$. For all figures $W_1 = 1$ and $W_2 = 0.8$.

are often active together). This phenomenon is shown in figure 1IIa,b. To better understand the qualitative behavior of our model we provide the following network interpretation (see figure 1IIa,b. Input to the model (i.e. the wavelet coefficients) undergo a nonlinear transformation $z_i \rightarrow |z_i|^{\beta_i} \rightarrow u = W|\mathbf{z}|^\beta \rightarrow u^\alpha$. The output of this network, $u^\alpha$, can be interpreted as a "penalty" for the input: the larger this penalty is, the more unlikely this input becomes under the probabilistic model. This process is most naturally understood [6] as enforcing constraints of the form $u = W|\mathbf{z}|^\beta \approx 0$, by penalizing violations of these constraints with $u^\alpha$.

What is the reason that the PoEdges model captures the clustering of wavelet activities? Consider a local model describing the statistical structure of a patch of wavelet coefficients and recall that the weighted sum of these activities is penalized. At a fixed position the activities are typically very small across images. However, when an edge happens to fall within the window of the model, most coefficients become active jointly. This "sparse" pattern of activity incurs less penalty than for instance the same amount[1] of activity distributed equally over all images because of the concave shape of the penalty function, i.e. $(\text{act})^\alpha < (\frac{1}{2}\text{act})^\alpha + (\frac{1}{2}\text{act})^\alpha$ where "act" is the activity level and $\alpha < 1$.

## 2.1 Related Work

Early wavelet denoising techniques were based on the observation that the marginal distribution of a wavelet coefficient is highly kurtotic (peaked and heavy tails). It was found that the generalized Gaussian density represents a very good fit to the empirical histograms [1, 7],

$$p(z) = \frac{\alpha w}{2\Gamma(\frac{1}{\alpha})} \exp\left[-(w|z|)^\alpha\right], \quad \alpha > 0, \; w > 0. \tag{1}$$

This has lead to the successful wavelet coring and shrinkage methods. A bivariate generalization of that model describing a wavelet coefficient $z_c$ and its "parent" $z_p$ at a higher level in the pyramid jointly, was proposed in [2]. The probability density,

$$p(z_c, z_p) = \frac{w}{2\pi} \exp\left(-\sqrt{w(z_c^2 + z_p^2)}\right) \tag{2}$$

is easily seen to be a special case of the PoEdges model proposed here. This model, unlike the univariate model, captures the bow-tie dependencies described above resulting a significant gain in denoising performance.

"Gaussian scale mixtures" (GSM) have been proposed to model even larger neighborhoods of wavelet coefficients. In particular, very good denoising results have been obtained by including within subband neighborhoods of size $3 \times 3$ in addition to the parent of a wavelet coefficient [3]. A GSM is defined in terms of a precision variable $u$, the square-root of which multiplies a multivariate Gaussian variable: $\mathbf{z} = \sqrt{u}\,\mathbf{y}$, $y \sim \mathcal{N}[0, \Sigma]$, resulting in the following expression for the distribution over the wavelet coefficients: $p(\mathbf{z}) = \int du\, \mathcal{N}_{\mathbf{z}}[0, u\Sigma]\, p(u)$. Here, $p(u)$ is the prior distribution for the precision variable. Hence, the GSM represents an example of a generative model with top-down semantics.

This in contrast to the PoEdges model which is better interpreted as a bottom-up network with log-probability proportional to its output. This difference is contrasted in figure 1IIa,b.

## 3 Edge-pert Denoising

Based on the PoEdges model discussed in the previous sections we now introduce a simplified model that forms the basis for a practical denoising algorithm. Recent progress in the field has indicated that it is important to model the higher order dependencies which exist between wavelet coefficients [2, 3]. This can be realized through the estimation of a *joint* model on a small cluster of wavelet coefficients around each coefficient. Ideally, we would like to use the full PoEdges model, but training these models from data is cumbersome. Therefore, in order to keep computations tractable, we proceed with a simplified model,

$$p(\mathbf{z}) \propto \exp\big[-\big(\sum_j w_j\big(\hat{\mathbf{a}}_j^T \mathbf{z}\big)^2\big)^\alpha\big]. \tag{3}$$

Compared to the full PoEdges model we use only one edge-pert and we have set $\beta_j = 2 \,\forall j$.

### 3.1 Model Estimation

Our next task is to estimate the parameters of this model efficiently. We will learn separate models for each wavelet coefficient jointly with a small neighborhood of dependent coefficients. Each such model is estimated in three steps: I) determine the coefficients that participate in each model, II) transform each model into a decorrelated domain (this implicitly estimates the $\{\hat{\mathbf{a}}_j\}$) and III) estimate the remaining parameters $\mathbf{w}, \alpha$ in the decorrelated domain using moment matching. Below we will describe these steps in more detail.

By $z_i, \tilde{z}_i$ we will denote the clean and noisy wavelet coefficients respectively. With $y_i, \tilde{y}_i$ we denote the *decorrelated* clean and noisy wavelet coefficients while $n_i$ denotes the Gaussian noise random variable in the wavelet domain, i.e. $\tilde{z}_i = z_i + n_i$. Both due to the details of the wavelet decomposition and due to the properties of the noise itself we assume the noise to be correlated and zero mean: $\mathbb{E}[n_i] = 0$, $\mathbb{E}[n_i n_j] = \Sigma_{ij}$. In this paper we further assume that we know the noise covariance in the image domain from which one can easily compute the noise covariance in the wavelet domain, however only minor changes are needed to estimate it from the noisy image itself.

**Step I**: We start with a $7 \times 7$ neighborhood from which we will adaptively select the best candidates to include in the model. In addition, we will always include the parent coefficient in the subband of a coarser scale if it exists (this is done by first up-sampling this band, see [3]). The coefficients that participate in a model are selected by estimating their dependencies relative to the center coefficient. Anticipating that (second order) correlations will be removed by sphering we are only interested in higher order dependencies, in particular dependencies between the variances. The following cumulant is used to obtain these estimates,

$$H_{cj} = \mathbb{E}[\tilde{z}_c^2 \tilde{z}_j^2] - 2\mathbb{E}[\tilde{z}_c \tilde{z}_j]^2 - \mathbb{E}[\tilde{z}_c^2]\mathbb{E}[\tilde{z}_j^2] \tag{4}$$

where $c$ is the center coefficient which will be denoised. The necessary averages $\mathbb{E}[\cdot]$ are computed by collecting samples within each subband, assuming that the statistics are location invariant. It can be shown that this cumulant is invariant under addition of possibly correlated Gaussian noise, i.e. it's value is the same for $\{z_i\}$ and $\{\tilde{z}_i\}$. Effectively, we measure the (higher order) dependencies between squared wavelet coefficients after subtraction of all correlations. Finally, we select the participants of a model centered at coefficient $\tilde{z}_c$ by ranking the positive $H_{cj}$ and picking all the ones which satisfy: $H_{ci} > 0.7 \times \max_{j \neq c} H_{cj}$.

**Step II:** For each model (with varying number of participants) we estimate the covariance,

$$C_{ij} = \mathbb{E}[z_i, z_j] = \mathbb{E}[\tilde{z}_i \tilde{z}_j] - \Sigma_{ij} \tag{5}$$

and correct it by setting to zero all negative eigenvalues in such a way that the sum of the eigenvalues is invariant (see [3]). Statistics are again collected by sampling within a subband. Then, we perform a linear transformation to a new basis onto which $\Sigma = \mathbf{I}$ and $C$ are diagonal. This can be accomplished by the following procedure,

$$RR^T = \Sigma \quad \Rightarrow \quad U\Lambda U^T = R^{-1}CR^{-T} \quad \Rightarrow \quad \tilde{\mathbf{y}} = (RU)^{-1}\tilde{\mathbf{z}}. \qquad (6)$$

In this new space (which is different for every wavelet coefficient) we can now assume $\hat{\mathbf{a}}_j = \mathbf{e}_j$, the axis aligned basis vector.

**Step III:** In the decorrelated space we estimate the single edge-pert model by moment matching. The moments of the edge-pert model in this space are easily computed using

$$\mathbb{E}\Big[(\sum_{j=1}^{N_p} w_j y_j^2)^\ell\Big] = \Gamma\Big(\frac{N_p + 2\ell}{2\alpha}\Big) / \ \Gamma\Big(\frac{N_p}{2\alpha}\Big) \qquad (7)$$

where $N_p$ is the number of participating coefficients in the model. We note that $\mathbb{E}[\tilde{y}_i^2] = 1 + \mathbb{E}[y_i^2]$. This leads to the following equation for $\alpha$

$$\frac{N_p^2 \Gamma\left(\frac{N_p+4}{2\alpha}\right)\Gamma\left(\frac{N_p}{2\alpha}\right)}{\Gamma\left(\frac{N_p+2}{2\alpha}\right)^2} = \sum_{i=1}^{N_p} \frac{\mathbb{E}[\tilde{y}_i^4] - 6\mathbb{E}[\tilde{y}_i^2] + 3}{(\mathbb{E}[\tilde{y}_i^2] - 1)^2} + \sum_{i\neq j}^{N_p} \frac{\mathbb{E}[\tilde{y}_i^2\tilde{y}_j^2] - \mathbb{E}[\tilde{y}_i^2] - \mathbb{E}[\tilde{y}_j^2] + 1}{(\mathbb{E}[\tilde{y}_i^2] - 1)(\mathbb{E}[\tilde{y}_j^2] - 1)}.$$

$$(8)$$

Thus we can estimate $\alpha$ by a line search and approximate the second term on the right hand side with $N_p(N_p - 1)$ to simplify the calculations. By further noting that the model (Eqn.3) is symmetric w.r.t. permutations of the variables $u_j = w_j y_j^2$ we find

$$w_j = \Gamma\big(\tfrac{N_p+2}{2\alpha}\big) / \ \Big(N_p(\mathbb{E}[\tilde{y}_i^2] - 1) \ \Gamma\big(\tfrac{N_p}{2\alpha}\big)\Big). \qquad (9)$$

A common strategy in the wavelet literature is to estimate the averages $\mathbb{E}[\cdot]$ by collecting samples in a local neighborhood around the coefficient under consideration. The advantage is that the estimates are adapting to the local statistics in the image. We have adopted this strategy and used a $11 \times 11$ box around each coefficient to collect 121 samples in the decorrelated wavelet domain. Coefficients for which $\mathbb{E}[\tilde{y}_i^2] < 1$ are set to zero and removed from consideration. The estimation of $\alpha$ depends on the fourth moment and is thus very sensitive to outliers, which is a commonly known problem with the moment matching method. We encounter the same problem so whenever we find no estimate of $\alpha$ in $[0, 1]$ using Eqn.8 we simply set it to $0.5$.

### 3.2 The Iterated Wiener Filter

To infer a wavelet coefficient given its noisy observation in the decorrelated wavelet domain, we maximize the *a posteriori* probability of our joint model. This is equivalent to,

$$\mathbf{z}^* = \underset{\mathbf{z}}{\operatorname{argmax}} \ \big( \log p(\tilde{\mathbf{z}}|\mathbf{z}) + \log p(\mathbf{z}) \big). \qquad (10)$$

When we assume Gaussian pixel noise, this translates into,

$$\mathbf{z}^* = \underset{\mathbf{z}}{\operatorname{argmin}} \ \Big( \tfrac{1}{2}(\mathbf{z} - \tilde{\mathbf{z}})^T K (\mathbf{z} - \tilde{\mathbf{z}}) + \big(\sum_j w_j z_j^2\big)^\alpha \Big) \qquad (11)$$

where $J$ is the (linear) wavelet transform $\tilde{\mathbf{z}} = J\mathbf{x}$, $K = J^{\#T}\Sigma_n^{-1}J^\#$ with $J^\# = (J^T J)^{-1}J^T$ the pseudo-inverse of $J$ (i.e. $J^\# J = I$) and $\Sigma_n$ the noise covariance matrix. In the decorrelated wavelet domain we simply set $K = \mathbf{I}$.

One can now construct an upper bound on this objective by using,

$$f^\alpha \leq \gamma f + (1 - \alpha)\big(\tfrac{\gamma}{\alpha}\big)^{\frac{\alpha}{\alpha-1}} \qquad \alpha < 1. \qquad (12)$$

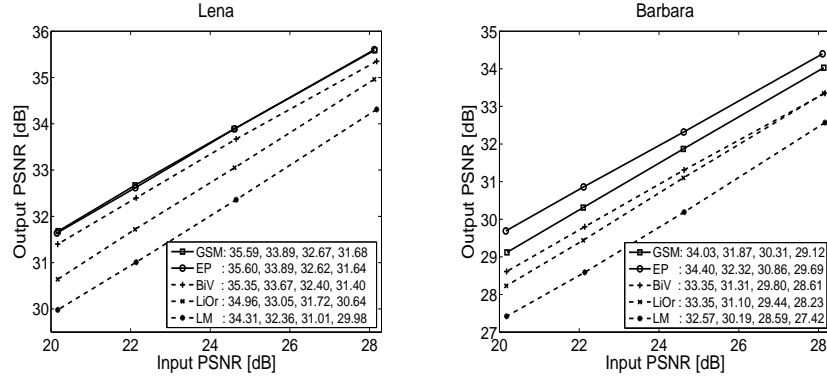

Figure 3: Output PSNR as a function of input PSNR for various methods on Lena (left) and Barbara (right) images. GSM: Gaussian scale mixture $(3 \times 3+p)$[3], EP: edge-pert, BIV: Bivariate adaptive shrinkage [2], LiOr: results from [8], LM: $5 \times 5$ LAWMAP results from [9]. Dashed lines indicate results copied from the literature, while solid lines indicate that the values were (re)produced on our computer.

This bound is saturated for $\gamma = \alpha f^{\alpha-1}$, and hence we can construct the following iterative algorithm that is guaranteed to converge to a local minimum,

$$\mathbf{z}^{t+1} = \left(K + \mathbf{Diag}[2\gamma^t \mathbf{w}]\right)^{-1} K\tilde{\mathbf{z}} \quad \Leftrightarrow \quad \gamma^{t+1} = \alpha\Big(\sum_j w_j(z_j^{t+1})^2\Big)^{\alpha-1}. \tag{13}$$

This algorithm has a natural interpretation as an "iterated Wiener filter" (IWF), since the first step (left hand side) is an ordinary Wiener filter while the second step (right hand side) adapts the variance of the filter. A summary of the complete algorithm is provided below.

---

## Edge-pert Denoising Algorithm

1. Decompose image into subbands.
2. *For each subband (except low-pass residual):*
2i.   Determine coefficients participating in joint model by using Eqn.4 (includes parent).
2ii.  Compute noise covariance $\Sigma$.
2iii. Compute signal covariance using Eqn.5.
3. *For each coefficient in a subband:*
3i.   Transform coefficients into the decorrelated domain using Eqn.6.
3ii.  Estimate parameters $\{\alpha, w_i\}$ on a local neighborhood using Eqn.8 and Eqn.9.
3iii. Denoise all wavelet coefficients in the neighborhood using IWF from section 3.2.
3iv.  Transform denoised cluster back to the wavelet domain and retain the "center coefficient" only.
4. Reconstruct denoised image by inverting the wavelet transform.

---

## 4   Experiments

Denoising experiments were run on the steerable wavelet pyramid with oriented high-pass residual bands (FSpyr) using 8 orientations as described in [3]. Results are reported on six images: "Lena", "Barbara", "Boat", "Fingerprint", "House" and "Peppers" and averaged over 5 experiments. In each experiment an image was artificially contaminated with independent Gaussian pixel noise of some predetermined variance and denoised using 20 iterations of the proposed algorithm. To reduce artifacts at the boundaries we used "reflective boundary extensions". The images were obtained from *http://decsai.ugr.es/~javier/denoise/index.html* to ensure comparison on the same set of images.

In table 1 we compare performance between the PoEdges and GSM based denoising algorithms on six test images and ten different noise levels. In figure 3 we compare results on

| $\sigma$ | | 1 | 2 | 5 | 10 | 15 | 20 | 25 | 50 | 75 | 100 |
|---|---|---|---|---|---|---|---|---|---|---|---|
| Lena | EP | **48.65** | **43.53** | **38.51** | 35.60 | 33.89 | 32.62 | 31.64 | 28.58 | 26.74 | 25.53 |
| | GSM | 48.46 | 43.23 | 38.49 | **35.61** | **33.90** | **32.66** | **31.69** | **28.61** | **26.84** | **25.64** |
| Barbara | EP | **48.70** | **43.59** | **38.06** | **34.40** | **32.32** | **30.86** | **29.69** | **26.12** | **24.12** | **22.90** |
| | GSM | 48.37 | 43.29 | 37.79 | 34.03 | 31.86 | 30.32 | 29.13 | 25.48 | 23.65 | 22.61 |
| Boat | EP | **48.46** | **43.09** | **37.05** | 33.49 | 31.58 | 30.28 | 29.24 | 26.27 | 24.64 | 23.56 |
| | GSM | 48.44 | 42.99 | 36.97 | **33.58** | **31.70** | **30.38** | **29.37** | **26.38** | **24.79** | **23.75** |
| Fingerprint | EP | 48.44 | 43.02 | 36.66 | 32.35 | 30.02 | 28.42 | 27.31 | 24.15 | **22.45** | **21.28** |
| | GSM | **48.46** | **43.05** | **36.68** | **32.45** | **30.14** | **28.60** | **27.45** | **24.16** | 22.40 | 21.22 |
| House | EP | **49.06** | **44.32** | **39.00** | **35.54** | **33.67** | 32.37 | 31.33 | 28.15 | 26.12 | 24.84 |
| | GSM | 48.85 | 44.07 | 38.65 | 35.35 | 33.64 | **32.39** | **31.40** | **28.26** | **26.41** | **25.11** |
| Peppers | EP | **48.50** | **43.20** | **37.40** | **33.79** | **31.74** | 30.29 | 29.13 | 25.69 | 23.85 | 22.50 |
| | GSM | 48.38 | 43.00 | 37.31 | 33.77 | **31.74** | **30.31** | **29.21** | **25.90** | **24.00** | **22.66** |

Table 1: Comparison of image denoising results between PoEdges (EP above) and its closest competitor (GSM). All results are averaged over 5 noise samples. The GSM results are copied from [3]. Details of the PoEdges algorithm are described in main text. Note that PoEdges outperforms GSM for low noise levels while the GSM performs better at high noise levels. Also, PoEdges performs best at all noise levels on the Barbara image, while GSM is superior on the boat image.

FSpyr against various methods published in the literature [3, 2, 9] on the images "Lena" and "Barbara".

These experiments lead to some interesting conclusions. In comparing PoEdges with GSM the general trend seems to be that PoEdges performs superior at lower noise levels while the reverse is true for higher noise levels. We observe that the PoEdges give significantly better results on the "Barbara" image than any other published method (by a large magin). According to the findings of the authors of [3][2] this stems mainly from the fact that the parameters are estimated locally which is particularly suited for this image. Increasing the estimation window in step 3ii of the algorithm let the denoising results drop down to the GSM solution (not reported here). Comparing the quality of restored images in detail (as in figure 3) we conclude that the GSM produces slightly sharper edges at the expense of more artifacts. Denoising a $512 \times 512$ pixel sized image on a pentium 4 $2.8GHz$ PC for our adaptive neighborhood selection model took 26 seconds for the QMF9 and $440$ seconds for the FSpyr.

We also compared GSM and EP using a separable orthonormal pyramid (QMF9). Using this simpler orthonormal decomposition we found that the EP model outperforms GSM in all experiments described above. However the results are significantly inferior because the wavelet representation plays a prominent role for denoising performance. These results and our matlab implementation of the algorithm are available online[3].

## 5   Discussion

We have proposed a general "product of edge-perts" model to capture the dependency structure in wavelet coefficients. This was turned into a practical denoising algorithm by simplifying to a single edge-pert and choosing $\beta_j = 2 \; \forall j$. The parameters of this model can be adapted based on the noisy observation of the image. In comparison with the closest competitor (GSM [3]) we found superior performance at low noise levels while the reverse is true for high noise levels. Also, the PoEdges model performs better than any competitor on the Barbara image, but consistency less well than GSM on the boat image.

The GSM model aims at capturing the same statistical regularities as the PoEdges but using a very different modelling paradigm: where PoEdges is best interpreted as a bottom-up constraint satisfaction model, the GSM is a causal generative model with top-down semantics. We have found that these two modelling paradigms exhibit different denoising accuracies

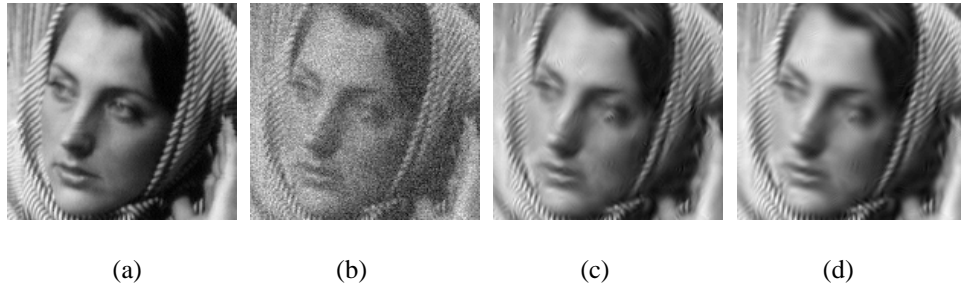

|  (a)  |  (b)  |  (c)  |  (d)  |

Figure 4: Comparison between (c) GSM with $3 \times 3$+parent [3] (PSNR 29.13) and (d) edge-pert denoiser with parameter settings as described in the text (PSNR 29.69) on Barbara image (cropped to $150 \times 150$ to enhance artifacts). Noisy image (b) has PSNR 20.17. Although the results turn out very similar, the GSM seems to be slightly less blurry at the expense of introducing more artifacts.

on some types of images implying an opportunity for further study and improvement.

The model in Eqn.3 can be extended in a number of ways. For example, we can lift the restriction on $\beta_j = 2$, allow more basis-vectors $\hat{\mathbf{a}}_j$ than coefficients or extend the neighborhood selection to subbands of different scales and/or orientations. More substantial performance gains are expected if we can extend the single edge-pert case to a multi edge-pert model. However, approximations in the estimation of these models will become necessary to keep the denoising algorithm practical. The adaptation of $\alpha$ relies on empirical estimations of the fourth moment and is therefore very sensitive to outliers. We are currently investigating more robust estimators to fit $\alpha$.

Further performance gains may still be expected through the development of new wavelet pyramids and through modelling of new dependency structures such as the phenomenon of phase alignment at the edges.

**Acknowledgments**   We would like to thank the authors of [2] and [3] for making their code available online.

## Footnotes

[1] We assume the total amount of variance in wavelet activity is fixed in this comparison.

[2]Personal communication

[3]http://www.kyb.mpg.de/~pgehler

# References

[1] J. Huang and D. Mumford. Statistics of natural images and models. In *Proc. of the Conf. on Computer Vision and Pattern Recognition*, pages 1541–1547, Ft. Collins, CO, USA, 1999.

[2] L. Sendur and I.W. Selesnick. Bivariate shrinkage with local variance estimation. *IEEE Signal Processing Letters*, 9(12):438–441, 2002.

[3] J. Portilla, V. Strela, M. Wainwright, and E. P. Simoncelli. Image denoising using scale mixtures of Gaussians in the wavelet domain. *IEEE Trans Image Processing*, 12(11):1338–1351, 2003.

[4] E.P. Simoncelli and W.T. Freeman. A flexible architecture for multi-scale derivative computation. In *IEEE Second Int'l Conf on Image Processing*, Washington DC, 1995.

[5] E.P. Simoncelli. Modeling the joint statistics of images in the wavelet domain. In *Proc SPIE, 44th Annual Meeting*, volume 3813, pages 188–195, Denver, 1999.

[6] G.E. Hinton and Y.W. Teh. Discovering multiple constraints that are frequently approximately satisfied. In *Proc. of the Conf. on Uncertainty in Artificial Intelligence*, pages 227–234, 2001.

[7] E.P. Simoncelli and E.H. Adelson. Noise removal via bayesian wavelet coring. In *3rd IEEE Int'l Conf on Image Processing*, Laussanne Switzerland, 1996.

[8] X. Li and M.T. Orchard. Spatially adaptive image denoising under over-complete expansion. In *IEEE Int'l. conf. on Image Processing*, Vancouver, BC, 2000.

[9] M. Kivanc, I. Kozintsev, K. Ramchandran, and P. Moulin. Low-complexity image denoising based on statistical modeling of wavelet coefficients. *IEEE Signal Proc. Letters*, 6:300–303, 1999.
